# Learning with Product Units

**Laurens R. Leerink**
Australian Gilt Securities LTD
37-49 Pitt Street
NSW 2000, Australia
laurens@sedal.su.oz.au

**C. Lee Giles**
NEC Research Institute
4 Independence Way
Princeton, NJ 08540, USA
giles@research.nj.nec.com

**Bill G. Horne**
NEC Research Institute
4 Independence Way
Princeton, NJ 08540, USA
horne@research.nj.nec.com

**Marwan A. Jabri**
Department of Electrical Engineering
The University of Sydney
NSW 2006, Australia
marwan@sedal.su.oz.au

## Abstract

Product units provide a method of automatically learning the higher-order input combinations required for efficient learning in neural networks. However, we show that problems are encountered when using backpropagation to train networks containing these units. This paper examines these problems, and proposes some atypical heuristics to improve learning. Using these heuristics a constructive method is introduced which solves well-researched problems with significantly less neurons than previously reported. Secondly, product units are implemented as candidate units in the Cascade Correlation (Fahlman & Lebiere, 1990) system. This resulted in smaller networks which trained faster than when using sigmoidal or Gaussian units.

## 1 Introduction

It is well-known that supplementing the inputs to a neural network with higher-order combinations of the inputs both increases the capacity of the network (Cover, 1965) and the the ability to learn geometrically invariant properties (Giles & Maxwell,

1987). However, there is a combinatorial explosion of higher order terms as the number of inputs to the network increases. Yet in order to implement a certain logical function, in most cases only a few of these higher order terms are required (Redding et al., 1993).

The product units (PUs) introduced by (Durbin & Rumelhart, 1989) attempt to make use of this fact. These networks have the advantage that, given an appropriate training algorithm, the units can automatically learn the higher order terms that are required to implement a specific logical function.

In these networks the hidden layer units compute the weighted product of the inputs, that is

$$\prod_{i=1}^{N} x_i^{w_i} \quad \text{instead of} \quad \sum_{i=1}^{N} x_i w_i \quad (1)$$

as in standard networks. An additional advantage of PUs is the increased information capacity of these units compared to standard summation networks. It is approximately $3N$ (Durbin & Rumelhart, 1989), compared to $2N$ for a single threshold logic function (Cover, 1965), where $N$ is the number of inputs to the unit.

The larger capacity means that the same functions can be implemented by networks containing less units. This is important for certain applications such as speech recognition where the data bandwidth is high or if realtime implementations are desired.

When PUs are used to process Boolean inputs, best performance is obtained (Durbin & Rumelhart, 1989) by using inputs of $\{+1, -1\}$. If the imaginary component is ignored, with these inputs, the activation function is equivalent to a cosine summation function with $\{-1, +1\}$ inputs mapped $\{1, 0\}$ (Durbin & Rumelhart, 1989). In the remainder of this paper the terms *product unit (PU)* and *cos(ine) unit* will be used interchangeably as all the problems examined have Boolean inputs.

## 2   Learning with Product Units

As the basic mechanism of a PU is multiplicative instead of additive, one would expect that standard neural network training methods and procedures cannot be directly applied when training these networks. This is indeed the case. If a neural network simulation environment is available the basic functionality of a PU can be obtained by simply adding the *cos* function $cos(\pi * input)$ to the existing list of transfer functions. This assumes that Boolean mappings are being implemented and the appropriate $\{-1, +1\} \rightarrow \{1, 0\}$ mapping has been performed on the input vectors. However, if we then attempt to train a network on on the parity-6 problem shown in (Durbin & Rumelhart, 1989), it is found that the standard backpropagation (BP) algorithm simply does not work. We have found two main reasons for this.

The first is weight initialization. A typical first step in the backpropagation procedure is to initialize all weights to small random values. The main reason for this is to use the dynamic range of the sigmoid function and it's derivative. However, the dynamic range of a PU is unlimited. Initializing the weights to small random

values results in an input to the unit where the derivative is small. So apart from choosing small weights centered around $n\pi$ with $n = \pm 1, \pm 2, \ldots$ this is the worst possible choice. In our simulations weights were initialized randomly in the range $[-2, 2]$. In fact, learning seems insensitive to the size of the weights, as long as they are large enough.

The second problem is local minima. Previous reports have mentioned this problem, (Lapedes & Farber, 1987) commented that "using *sin*'s often leads to numerical problems, and nonglobal minima, whereas sigmoids seemed to avoid such problems". This comment summarizes our experience of training with PUs. For small problems (less than 3 inputs) backpropagation provides satisfactory training. However, when the number of inputs are increased beyond this number, even with the weight initialization in the correct range, training usually ends up in a local minima.

## 3   Training Algorithms

With these aspects in mind, the following training algorithms were evaluated: online and batch versions of Backpropagation (BP), Simulated Annealing (SA), a Random Search Algorithm (RSA) and combinations of these algorithms.

BP was used as a benchmark and for use in combination with the other algorithms. The Delta-Bar-Delta learning rate adaptation rule (Jacobs, 1988) was used along with the batch version of BP to accelerate convergence, with the parameters were set to $\theta = 0.35, \kappa = 0.05$ and $\phi = 0.90$. RSA is a global search method (i.e. the whole weight space is explored during training). Weights are randomly chosen from a predefined distribution, and replaced if this results in an error decrease. SA (Kirkpatrick et al., 1983) is a standard optimization method. The operation of SA is similar to RSA, with the difference that with a decreasing probability solutions are accepted which increase the training error. The combination of algorithms were chosen (BP & SA, BP & RSA) to combine the benefits of global and local search. Used in this manner, BP is used to find the local minima. If the training error at the minima is sufficiently low, training is terminated. Otherwise, the global method initializes the weights to another position in weight space from which local training can continue.

The BP-RSA combination requires further explanation. Several BP-(R)SA combinations were evaluated, but best performance was obtained using a fixed number of iterations of BP (in this case 120) along with one initial iteration of RSA. In this manner BP is used to move to the local minima, and if the training error is still above the desired level the RSA algorithm generates a new set of random weights from which BP can start again.

The algorithms were evaluated on two problems, the parity problem and learning all logical functions of 2 and 3 inputs. The infamous parity problem is (for the product unit at least) an appropriate task. As illustrated by (Durbin & Rumelhart, 1989), this problem can be solved by one product unit. The question is whether the training algorithms can find a solution. The target values are $\{-1, +1\}$, and the output is taken to be correct if it has the correct sign. The simulation results are shown in Table 1. It should be noted that one epoch of both SA and RSA involves

relaxing the network across the training set for every weight, so in computational terms their $\overline{n}_{epoch}$ values should be multiplied by a factor of $(N+1)$.

| Parity | Online BP | | Batch BP | | SA | | RSA | |
|---|---|---|---|---|---|---|---|---|
| $N$ | $n_{conv}$ | $\overline{n}_{epoch}$ | $n_{conv}$ | $\overline{n}_{epoch}$ | $n_{conv}$ | $\overline{n}_{epoch}$ | $n_{conv}$ | $\overline{n}_{epoch}$ |
| 6 | 10 | 30.4 | 7 | 34 | 10 | 12.6 | 10 | 15.2 |
| 8 | 8 | 101.3 | 2 | 700 | 10 | 52.8 | 10 | 45.4 |
| 10 | 6 | 203.3 | 0 | - | 10 | 99.9 | 10 | 74.1 |

Table 1: The parity $N$ problem: The table shows $n_{conv}$ the number of runs out of 10 that have converged and $\overline{n}_{epoch}$, the average number of training epochs required when training converged.

For the parity problem it is clear that local learning alone does not provide good convergence. For this problem, global search algorithms have the following advantages: (1) The search space is bounded (all weights are restricted to $[-2, +2]$) (2) The dimension of search space is low (maximum of 11 weights for the problems examined). (3) The fraction of the weight space which satisfies the parity problem relative to the total bounded weight space is high.

In a second set of simulations, one product unit was trained to calculate all $2^{(2^N)}$ logical functions of the $N$ input variables. Unfortunately, this is only practical for $N \in \{2, 3\}$. For $N = 2$ there are only 16 functions, and a product unit has no problem learning all these functions rapidly with all four training algorithms. In comparison a single summation unit can learn 14 (not the XOR & XNOR functions). For $N=3$, a product unit is able to implement 208 of the 256 functions, while a single summation unit could only implement 104. The simulation results are displayed in Table 2.

| Online BP | | Batch BP | | SA | | RSA | | BP-RSA | |
|---|---|---|---|---|---|---|---|---|---|
| $\overline{n}_{logic}$ | $\overline{n}_{epoch}$ | $\overline{n}_{logic}$ | $\overline{n}_{epoch}$ | $\overline{n}_{logic}$ | $\overline{n}_{epoch}$ | $\overline{n}_{logic}$ | $\overline{n}_{epoch}$ | $\overline{n}_{logic}$ | $\overline{n}_{epoch}$ |
| 147.3 | 42.6 | 189.2 | 20.5 | 196.1 | 43.8 | 167.4 | 60.2 | 208 | 44.3 |

Table 2: Learning all logical functions of 3 inputs: The rows display $\overline{n}_{logic}$, the average number of logical functions implemented by a product unit and $\overline{n}_{epoch}$, the number of epochs required for convergence. Ten simulations were performed for each of the 256 logical functions, each for a maximum of 1,000 iterations.

## 4   Constructive Learning with Product Units

Selecting the optimal network architecture for a specific application is a nontrivial and time-consuming task, and several algorithms have been proposed to automate this process. These include pruning methods and growing algorithms. In this section a simple method is proposed for adding PUs to the hidden layer of a three layer network. The output layer contains a single sigmoidal unit.

Several constructive algorithms proceed by freezing a subset of the weights and limiting training to the newly added units. As mentioned earlier, for PUs a global

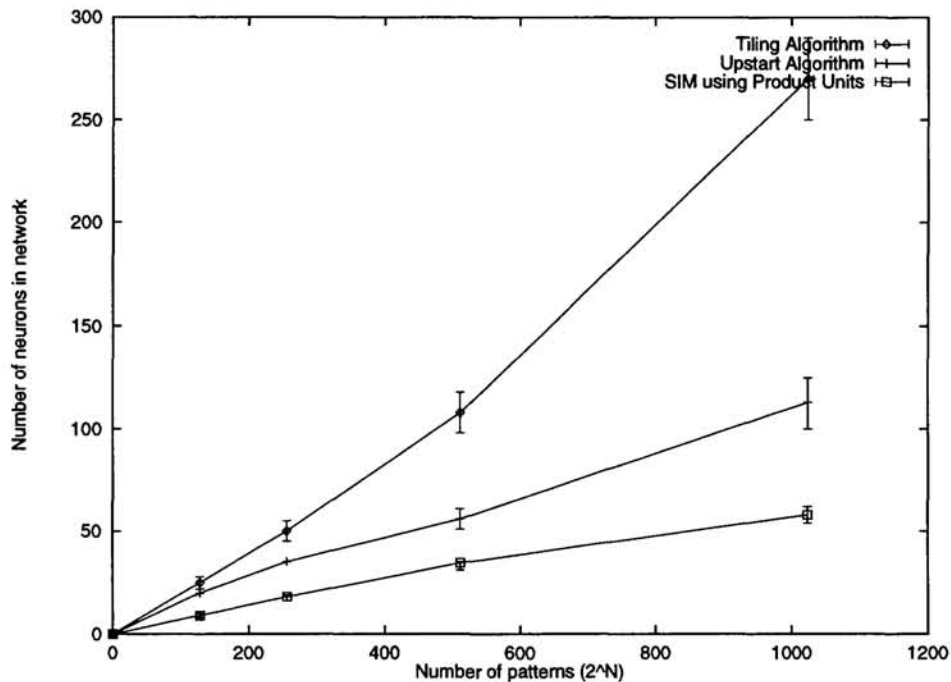

Figure 1: The number of units required for learning the random mapping problems by the 'Tiling', 'Upstart' and SIM algorithms.

search is required to solve the local-minima problems. Freezing a subset of the weights restricts the new solution to an affine subset of the existing weight space, often resulting in non-minimal networks (Ash, 1989). For this reason a simple incremental method (SIM) was implemented which retains the global search for all weights during the whole training process. The method used in our simulations is as follows:

- Train a network using the BP-RSA combination on a network with a specified minimum number of hidden PUs.

- If there is no convergence within a specified number of epochs, add a PU to the network. Reinitialize weights and continue training with the BP-RSA combination.

- Repeat process until a solution is found or the network has grown a predetermined maximum size.

The method of (Ash, 1989) was also evaluated, where neurons with small weights were added to a network according to certain criteria. The SIM performed better, possibly because of the global search performed by the RSA step.

The 'Upstart' (Frean, 1990) and 'Tiling' (Mézard & Nadal, 1989) constructive algorithms were chosen as benchmarks. A constructive PU network was trained on two problems described in these papers, namely the parity problem and the random mapping problem. In (Frean, 1990) it was reported that the Upstart

algorithm required $N$ units for all parity $N$ problems, and 1,000 training epochs were sufficient for all values of $N$ except $N = 10$, which required 10,000. As seen earlier, one PU is able to perform any parity function, and SIM required an an average of 74.1 iterations for $N = 6, 8, 10$.

The random mapping problem is defined by assigning each of the $2^N$ patterns its target $\{-1, +1\}$ with 50% probability. This is a difficult problem, due to the absence of correlations and structure in the input. As in (Frean, 1990; Mézard & Nadal, 1989) the average of 25 runs were performed, each on a different training set. The number of units required by SIM is plotted in Figure 1. The values for the Tiling and Upstart algorithms are approximate and were obtained through inspection from a similar graph in (Frean, 1990).

## 5    Using Cosine Candidate Units in Cascade Correlation

Initially we wanted to compare the performance of SIM with the well-known 'cascade-correlation' (CC) algorithm of (Fahlman & Lebiere, 1990). However, the network architectures differ and a direct comparison between the number of units in the respective architectures does not reflect the efficiency of the algorithms. Instead, it was decided to integrate PUs into the CC system as candidate units.

For these simulations a public domain version of CC was used (White, 1993) which supports four different candidate types; the asymmetric sigmoid, symmetric sigmoid, variable sigmoid and gaussian units. Facilities exist for either constructing homogeneous networks by selecting one unit type, or training with a pool of different units allowing the construction of hybrid networks. It was thus relatively simple to add PU candidate units to the system. Table 3 displays the results when CC was trained on the random logic problem using three types of homogeneous candidate units.

| $N$ | CC Sigmoid | | CC Gauss | | CC PU | |
|---|---|---|---|---|---|---|
| | $\overline{n}_{units}$ | $\overline{n}_{epochs}$ | $\overline{n}_{units}$ | $\overline{n}_{epochs}$ | $\overline{n}_{units}$ | $\overline{n}_{epochs}$ |
| 7 | 6.6 | 924.5 | 6.7 | 642.6 | 5.7 | 493.8 |
| 8 | 12.1 | 1630.9 | 11.5 | 1128.2 | 9.9 | 833.8 |
| 9 | 20.5 | 2738.3 | 18.4 | 1831.1 | 16.4 | 1481.8 |
| 10 | 32.9 | 4410.9 | 30.2 | 2967.6 | 26.6 | 2590.8 |

Table 3: Learning random logic functions of $N$ inputs: The table shows $\overline{n}_{units}$, the average number of units required and $\overline{n}_{epochs}$, the average number of training epochs required for convergence of CC using sigmoidal, Gaussian and PU candidate units. Figures are based on 25 simulations.

In a separate experiment the performance of hybrid networks were re-evaluated on the same random logic problem. To enable a fair competition between candidate units of different types, the simulations were run with 40 candidate units, 8 of each type. The simulations were evaluated on 25 trails for each of the random mapping problems (7,8,9 and 10 inputs, a total of 1920 input vectors). In total 1460 hidden units were allocated, and in *all cases* PU candidate units were chosen above units of the 4 other types during the competitive stage. During this comparison all

parameters were set to default values, i.e. the weights of the PU candidate units were random numbers initialized in the range of $[-1, +1]$. As discussed earlier, this puts the PUs at a slight disadvantage as their optimum range is $[-2, +2]$.

## 6   Discussion

The BP-RSA combination is in effect equivalent to the 'local optimization with random restarts' process discussed by (Karmarkar & Karp, 1982), where the local optimization is this case is performed by the BP algorithm. They reported that for certain problems where the error surface was 'exceedingly mountainous', multiple random-start local optimization outperformed more sophisticated methods. We hypothesize that adding PUs to a network makes the error surface sufficiently mountainous so that a global search is required.

As expected, the higher separating capacity of the PU enables the construction of networks with less neurons than those produced by the Tiling and Upstart algorithms. The fact that SIM works this well is mainly a result of the error surface; the surface is so irregular that even training a network of fixed architecture is best done by reinitializing the weights if convergence does not occur within certain bounds. This again is in accordance with the results of (Karmarkar & Karp, 1982) discussed above.

When used in CC we hypothesize that there are three main reasons for the choice of PUs above any of the other types during the competitive learning phase. Firstly, the higher capacity (in a information capacity sense) of the PUs allows a better correlation with the error signal. Secondly, having $N$ competing candidate units is equivalent to selecting the best of $N$ random restarts, and performs the required global search. Thirdly, although the error surface of networks with PUs contains more local minima than when using standard transfer functions, the surface is locally smooth. This allows effective use of higher-order error derivatives, resulting in fast convergence by the quickprop algorithm.

In (Dawson & Schopflocher, 1992) it was shown that networks with Gaussian units train faster and require less units than networks with standard sigmoidal units. This is supported by our results shown in Table 3. However, for the problem examined, PUs outperform Gaussian units by approximately the same margin as Gaussian units outperform sigmoidal units. It should also be noted that these problems where not chosen for their suitability for PUs. In fact, if the problems are symmetric/regular the difference in performance is expected to increase.

## 7   Conclusion

Of the learning algorithms examined BP provides the fastest training, but is prone to nonglobal minima. On the other hand, global search methods are impractical for larger networks. For the problems examined, a combination of local and global search methods were found to perform best. Given a network containing PUs, there are some atypical heuristics that can be used: (a) correct weight initialization (b) reinitialization of the weights if convergence is not rapidly reached. In addition, the representational power of PUs have enabled us to solve standard problems

using significantly smaller networks than previously reported, using a very simple constructive method. When implemented in the CC architecture, for the problems examined PUs resulted in smaller networks which trained faster than other units. When included in a pool of competing candidate units, simulations showed that in all cases PU candidate units were preferred over candidate units of the other four types.

# References

Ash, T. (1989). Dynamic node creation in backpropagation networks. *Connection Science, 1*(4), 365–375.

Cover, T. (1965). Geometrical and statistical properties of systems of linear inequalities with applications in pattern recognition. *IEEE Transactions on Electronic Computers, 14*, 326–334.

Dawson, M. & Schopflocher, D. (1992). Modifying the generalized delta rule to train networks of nonmonotonic processors for pattern classification. *Connection Science, 4*, 19–31.

Durbin, R. & Rumelhart, D. (1989). Product units: A computationally powerful and biologically plausible extension to backpropagation networks. *Neural Computation, 1*, 133–142.

Fahlman, S. & Lebiere, C. (1990). The cascade-correlation learning architecture. In Touretzky, D. (Ed.), *Advances in Neural Information Processing Systems*, volume 2, (pp. 524–532)., San Mateo. (Denver 1989), Morgan Kaufmann.

Frean, M. (1990). The upstart algorithm: A method for constructing and training feedforward neural networks. *Neural Computation, 2*, 198–209.

Giles, C. & Maxwell, T. (1987). Learning, invariance, and generalization in high-order neural networks. *Applied Optics, 26*(23), 4972–4978.

Jacobs, R. (1988). Increased rates of convergence through learning rate adaptation. *Neural Networks, 1*, 295–307.

Karmarkar, N. & Karp, R. (1982). The differencing method of set partitioning. Technical Report UCB/CSD 82/113, Computer Science Division, University of California, Berkeley, California.

Kirkpatrick, S., Jr., C. G., , & Vecchi, M. (1983). Optimization by simulated annealing. *Science, 220*. Reprinted in (?).

Lapedes, A. & Farber, R. (1987). Nonlinear signal processing using neural networks: Prediction and system modelling. Technical Report LA–UR–87–2662, Los Alamos National Laboratory, Los Alamos, NM.

Mézard, M. & Nadal, J.-P. (1989). Learning in feedforward layered networks: The tiling algorithm. *Journal of Physics A, 22*, 2191–2204.

Redding, N., Kowalczyk, A., & Downs, T. (1993). A constructive higher order network algorithm that is polynomial-time. *Neural Networks, 6*, 997.

White, M. (1993). A public domain C implemention of the Cascade Correlation algorithm. Department of Computer Science, Carnegie Mellon University, Pittsburgh, PA.